# Saliency Based on Information Maximization

**Neil D.B. Bruce and John K. Tsotsos**
Department of Computer Science and Centre for Vision Research
York University, Toronto, ON, M2N 5X8
{neil,tsotsos}@cs.yorku.ca

## Abstract

A model of bottom-up overt attention is proposed based on the principle of maximizing information sampled from a scene. The proposed operation is based on Shannon's self-information measure and is achieved in a neural circuit, which is demonstrated as having close ties with the circuitry existent in the primate visual cortex. It is further shown that the proposed saliency measure may be extended to address issues that currently elude explanation in the domain of saliency based models. Results on natural images are compared with experimental eye tracking data revealing the efficacy of the model in predicting the deployment of overt attention as compared with existing efforts.

## 1 Introduction

There has long been interest in the nature of eye movements and fixation behavior following early studies by Buswell [1] and Yarbus [2]. However, a complete description of the mechanisms underlying these peculiar fixation patterns remains elusive. This is further complicated by the fact that task demands and contextual knowledge factor heavily in how sampling of visual content proceeds.

Current bottom-up models of attention posit that *saliency* is the impetus for selection of fixation points. Each model differs in its definition of saliency. In perhaps the most popular model of bottom-up attention, saliency is based on centre-surround contrast of units modeled on known properties of primary visual cortical cells [3]. In other efforts, saliency is defined by more *ad hoc* quantities having less connection to biology [4]. In this paper, we explore the notion that *information* is the driving force behind attentive sampling.

The application of information theory in this context is not in itself novel. There exist several previous efforts that define saliency based on Shannon entropy of image content defined on a local neighborhood [5, 6, 7, 8]. The model presented in this work is based on the closely related quantity of self-information [9]. In section 2.2 we discuss differences between entropy and self-information in this context, including why self-information may present a more appropriate metric than entropy in this domain. That said, contributions of this paper are as follows:

1. A bottom-up model of overt attention with selection based on the self-information of local image content.

2. A qualitative and quantitative comparison of predictions of the model with human

eye tracking data, contrasted against the model of Itti and Koch [3].

3. Demonstration that the model is neurally plausible via implementation based on a neural circuit resembling circuitry involved in early visual processing in primates.

4. Discussion of how the proposal generalizes to address issues that deny explanation by existing saliency based attention models.

## 2 The Proposed Saliency Measure

There exists much evidence indicating that the primate visual system is built on the principle of establishing a sparse representation of image statistics. In the most prominent of such studies, it was demonstrated that learning a sparse code for natural image statistics results in the emergence of simple-cell receptive fields similar to those appearing in the primary visual cortex of primates [10, 11]. The apparent benefit of such a representation comes from the fact that a sparse representation allows certain independence assumptions with regard to neural firing. This issue becomes important in evaluating the likelihood of a set of local image statistics and is elaborated on later in this section.

In this paper, saliency is determined by quantifying the self-information of each local image patch. Even for a very small image patch, the probability distribution resides in a very high dimensional space. There is insufficient data in a single image to produce a reasonable estimate of the probability distribution. For this reason, a representation based on independent components is employed for the independence assumption it affords. ICA is performed on a large sample of 7x7 RGB patches drawn from natural images to determine a suitable basis. For a given image, an estimate of the distribution of each basis coefficient is learned across the entire image through non-parametric density estimation. The probability of observing the RGB values corresponding to a patch centred at any image location may then be evaluated by independently considering the likelihood of each corresponding basis coefficient. The product of such likelihoods yields the joint likelihood of the entire set of basis coefficients. Given the basis determined by ICA, the preceding computation may be realized entirely in the context of a biologically plausible neural circuit. The overall architecture is depicted in figure 1. Details of each of the aforesaid model components including the details of the neural circuit are as follows:

Projection into independent component space provides, for each local neighborhood of the image, a vector $w$ consisting of $N$ variables $w_i$ with values $v_i$. Each $w_i$ specifies the contribution of a particular basis function to the representation of the local neighborhood. As mentioned, these basis functions, learned from statistical regularities observed in a large set of natural images show remarkable similarity to V1 cells [10, 11]. The ICA projection then allows a representation $w$, in which the components $w_i$ are as independent as possible. For further details on the ICA projection of local image statistics see [12]. In this paper, we propose that salience may be defined based on a strategy for maximum information sampling. In particular, Shannon's self-information measure [9], $-log(p(x))$, applied to the joint likelihood of statistics in a local neighborhood decribed by $w$, provides an appropriate transformation between probability and the degree of information inherent in the local statistics. It is in computing the observation likelihood that a sparse representation is instrumental: Consider the probability density function $p(w_1 = v_1, w_2 = v_2, ..., w_n = v_n)$ which quantifies the likelihood of observing the local statistics with values $v_1, ..., v_n$ within a particular context. An appropriate context may include a larger area encompassing the local neigbourhood described by $w$, or the entire scene in question. The presumed independence of the ICA decomposition means that $p(w_1 = v_1, w_2 = v_2, ..., w_n = v_n) = \prod_{i=1}^{n} p(w_i = v_i)$. Thus, a sparse representation allows the estimation of the $n$-dimensional space described by $w$ to be derived from $n$ one dimensional probability density functions. Evaluating $p(w_1 = v_1, w_2 = v_2, ..., w_n = v_n)$ requires considering the distribution of values taken on by each $w_i$ in a more global context. In practice, this might be derived on the basis of a

nonparametric or histogram density estimate. In the section that follows, we demonstrate that an operation equivalent to a non-parametric density estimate may be achieved using a suitable neural circuit.

## 2.1 Likelihood Estimation in A Neural Circuit

In the following formulation, we assume an estimate of the likelihood of the components of $w$ based on a Gaussian kernel density estimate. Any other choice of kernel may be substituted, with a Gaussian window chosen only for its common use in density estimation and without loss of generality.

Let $w_{i,j,k}$ denote the set of independent coefficients based on the neighborhood centered at $j, k$. An estimate of $p(w_{i,j,k} = v_{i,j,k})$ based on a Gaussian window is given by:

$$\frac{1}{\sigma\sqrt{2\pi}} \sum_{\forall s,t \in \Psi} \omega(s,t) e^{-(v_{i,j,k}-v_{i,s,t})^2/2\sigma^2} \tag{1}$$

with $\sum_{s,t} \omega(s,t) = 1$ where $\Psi$ is the context on which the probability estimate of the coefficients of $\omega$ is based. $\omega(s,t)$ describes the degree to which the coefficient $\omega$ at coordinates $s, t$ contributes to the probability estimate. On the basis of the form given in equation 1 it is evident that this operation may equivalently be implemented by the neural circuit depicted in figure 2. Figure 2 demonstrates only coefficients derived from a horizontal cross-section. The two dimensional case is analogous with parameters varying in $i, j$, and $k$ dimensions. $K$ consists of the Kernel function employed for density estimation. In our case this is a Gaussian of the form $\frac{1}{\sigma\sqrt{2\pi}}e^{-x^2/2\sigma^2}$. $\omega(s,t)$ is encoded based on the weight of connections to $K$. As $x = v_{i,j,k} - v_{i,s,t}$ the output of this operation encodes the impact of the Kernel function with mean $v_{i,s,t}$ on the value of $p(w_{i,j,k} = v_{i,j,k})$. Coefficients at the input layer correspond to coefficients of $v$. The logarithmic operator at the final stage might also be placed before the product on each incoming connection, with the product then becoming a summation. It is interesting to note that the structure of this circuit at the level of within feature spatial competition is remarkably similar to the standard feedforward model of lateral inhibition, a ubiquitous operation along the visual pathways thought to play a chief role in attentional processing [14]. The similarity between independent components and V1 cells, in conjunction with the aforementioned consideration lends credibility to the proposal that *information* may contribute to driving overt attentional selection.

One aspect lacking from the preceding description is that the saliency map fails to take into account the dropoff in visual acuity moving peripherally from the fovea. In some instances the maximum information accommodating for visual acuity may correspond to the center of a cluster of salient items, rather than centered on one such item. For this reason, the resulting saliency map is convolved with a Gaussian with parameters chosen to correspond approximately to the dropoff in visual acuity observed in the human visual system.

## 2.2 Self-Information versus Entropy

It is important to distinguish between self-information and entropy since these terms are often confused. The difference is subtle but important on two fronts. The first consideration lies in the expected behavior in *popout* paradigms and the second in the neural circuitry involved.

Let $X = [x_1, x_2, ..., x_n]$ denote a vector of RGB values corresponding to image patch $X$, and $D$ a probability density function describing the distribution of some feature set over $X$. For example, $D$ might correspond to a histogram estimate of intensity values within $X$ or the relative contribution of different orientations within a local neighborhood situated on the boundary of an object silhouette [6]. Assuming an estimate of $D$ based on $N$

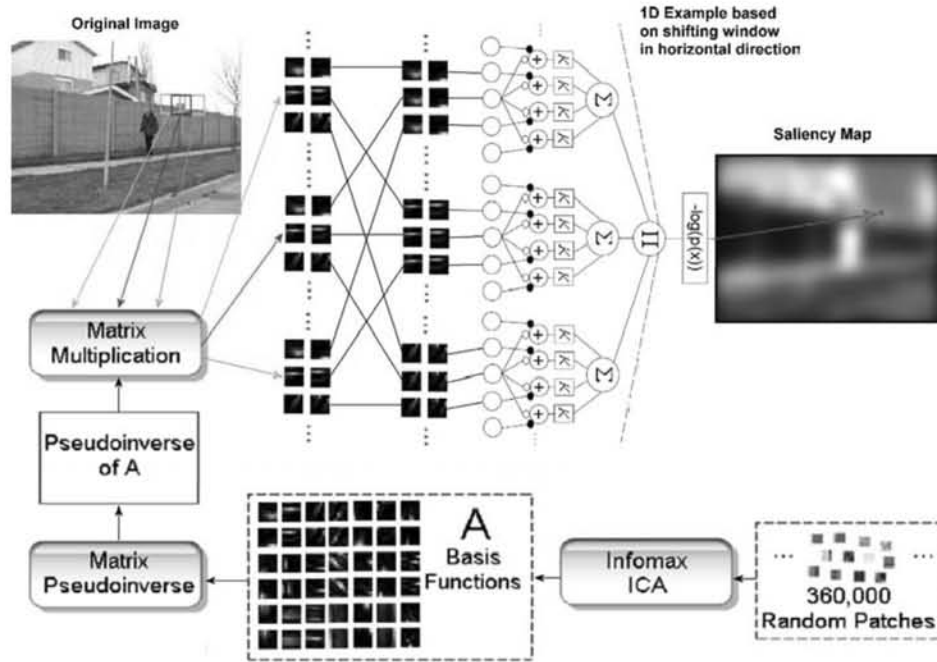

Figure 1: The framework that achieves the desired information measure. Shown is the computation corresponding to three horizontally adjacent neighbourhoods with flow through the network indicated by the orange, purple, and cyan windows and connections. The connections shown facilitate computation of the information measure corresponding to the pixel centered in the purple window. The network architecture produces this measure on the basis of evaluating the probability of these coefficients with consideration to the values of such coefficients in neighbouring regions.

bins, the entropy of D is given by: $-\sum_{i=1}^{N} D_i log(D_i)$. In this example, entropy characterizes the extent to which the feature(s) characterized by $D$ are *uniformly* distributed on $X$. Self-information in the proposed saliency measure is given by $-log(p(X))$. That is, Self-information characterizes the raw likelihood of the specific n-dimensional vector of RGB values given by $X$. $p(X)$ in this case is based on observing a number of n-dimensional feature vectors based on patches drawn from the area surrounding $X$. Thus, $p(X)$ characterizes the raw likelihood of observing $X$ based on its surround and $-log(p(X))$ becomes closer to a measure of local contrast whereas entropy as defined in the usual manner is closer to a measure of local activity. The importance of this distinction is evident in considering figure 3. Figure 3 depicts a variety of candles of varying orientation, and color. There is a tendency to fixate the empty region on the left, which is the location of lowest entropy in the image. In contrast, this region receives the highest confidence from the algorithm proposed in this paper as it is highly informative in the context of this image. In classic popout experiments, a vertical line among horizontal lines presents a highly salient target. The same vertical line among many lines of random orientations is not, although the entropy associated with the second scenario is much greater.

With regard to the neural circuitry involved, we have demonstrated that self-information may be computed using a neural circuit in the absence of a representation of the entire probability distribution. Whether an equivalent operation may be achieved in a biologically plausible manner for the computation of entropy remains to be established.

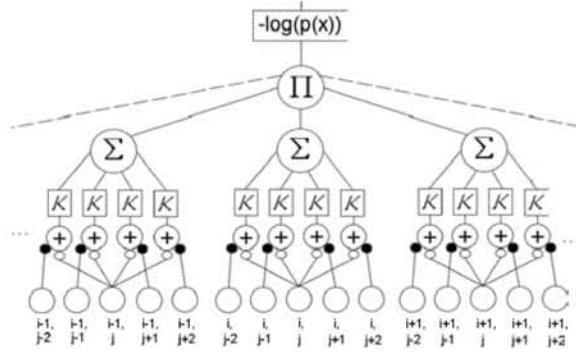

Figure 2: A 1D depiction of the neural architecture that computes the self-information of a set of local statistics. The operation is equivalent to a Kernel density estimate. Coefficients correspond to subscripts of $v_{i,j,k}$. The small black circles indicate an inhibitory relationship and the small white circles an excitatory relationship

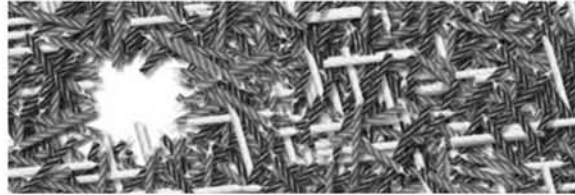

Figure 3: An image that highlights the difference between entropy and self-information. Fixation invariably falls on the empty patch, the locus of minimum entropy in orientation and color but maximum in self-information when the surrounding context is considered.

## 3 Experimental Validation

The following section evaluates the output of the proposed algorithm as compared with the bottom-up model of Itti and Koch [3]. The model of Itti and Koch is perhaps the most popular model of saliency based attention and currently appears to be the yardstick against which other models are measured.

### 3.1 Experimental eye tracking data

The data that forms the basis for performance evaluation is derived from eye tracking experiments performed while subjects observed 120 different color images. Images were presented in random order for 4 seconds each with a mask between each pair of images. Subjects were positioned 0.75m from a 21 inch CRT monitor and given no particular instructions except to observe the images. Images consist of a variety of indoor and outdoor scenes, some with very salient items, others with no particular regions of interest. The eye tracking apparatus consisted of a standard non head-mounted device. The parameters of the setup are intended to quantify salience in a general sense based on stimuli that one might expect to encounter in a typical urban environment. Data was collected from 20 different subjects for the full set of 120 images.

The issue of comparing between the output of a particular algorithm, and the eye tracking data is non-trivial. Previous efforts have selected a number of fixation points based on the saliency map, and compared these with the experimental fixation points derived

from a small number of subjects and images (7 subjects and 15 images in a recent effort [4]). There are a variety of methodological issues associated with such a representation. The most important such consideration is that the representation of perceptual importance is typically based on a saliency map. Observing the output of an algorithm that selects fixation points based on the underlying saliency map obscures observation of the degree to which the saliency maps predict important and unimportant content and in particular, ignores confidence away from highly salient regions. Secondly, it is not clear how many fixation points should be selected. Choosing this value based on the experimental data will bias output based on information pertaining to the content of the image and may produce artificially good results.

The preceding discussion is intended to motivate the fact that selecting discrete fixation co-ordinates based on the saliency map for comparison may not present the most appropriate representation to use for performance evaluation. In this effort, we consider two different measures of performance. Qualitative comparison is based on the representation proposed in [16]. In this representation, a fixation density map is produced for each image based on all fixation points, and subjects. Given a fixation point, one might consider how the image under consideration is sampled by the human visual system as photoreceptor density drops steeply moving peripherally from the centre of the fovea. This dropoff may be modeled based on a 2D Gaussian distribution with appropriately chosen parameters, and centred on the measured fixation point. A continuous fixation density map may be derived for a particular image based on the sum of all 2D Gaussians corresponding to each fixation point, from each subject. The density map then comprises a measure of the extent to which each pixel of the image is sampled on average by a human observer based on observed fixations. This affords a representation for which similarity to a saliency map may be considered at a glance. Quantitative performance evaluation is achieved based on the measure proposed in [15]. The saliency maps produced by each algorithm are treated as binary classifiers for fixation versus non-fixation points. The choice of several different thresholds and assessment of performance in predicting fixated versus not fixated pixel locations allows an ROC curve to be produced for each algorithm.

### 3.2 Experimental Results

Figure 4 affords a qualitative comparison of the output of the proposed model with the experimental eye tracking data for a variety of images. Also depicted is the output of the Itti and Koch algorithm for comparison.

In the implementation results shown, the ICA basis set was learned from a set of 360,000 7x7x3 image patches from 3600 natural images using the Lee et al. extended infomax algorithm [17]. Processed images are 340 by 255 pixels. $\Psi$ consists of the entire extent of the image and $\omega(s,t) = \frac{1}{p} \forall s,t$ with $p$ the number of pixels in the image. One might make a variety of selections for these variables based on arguments related to the human visual system, or based on performance. In our case, the values have been chosen on the basis of simplicity and do not appear to dramatically affect the predictive capacity of the model in the simulation results. In particular, we wished to avoid tuning these parameters to the available data set. Future work may include a closer look at some of the parameters involved in order to determine the most appropriate choices. The ROC curves appearing in figure 5 give some sense of the efficacy of the model in predicting which regions of a scene human observers tend to fixate. As may be observed, the predictive capacity of the model is on par with the approach of Itti and Koch. Encouraging is the fact that similar performance is achieved using a method derived from first principles, and with no parameter tuning or *ad hoc* design choices.

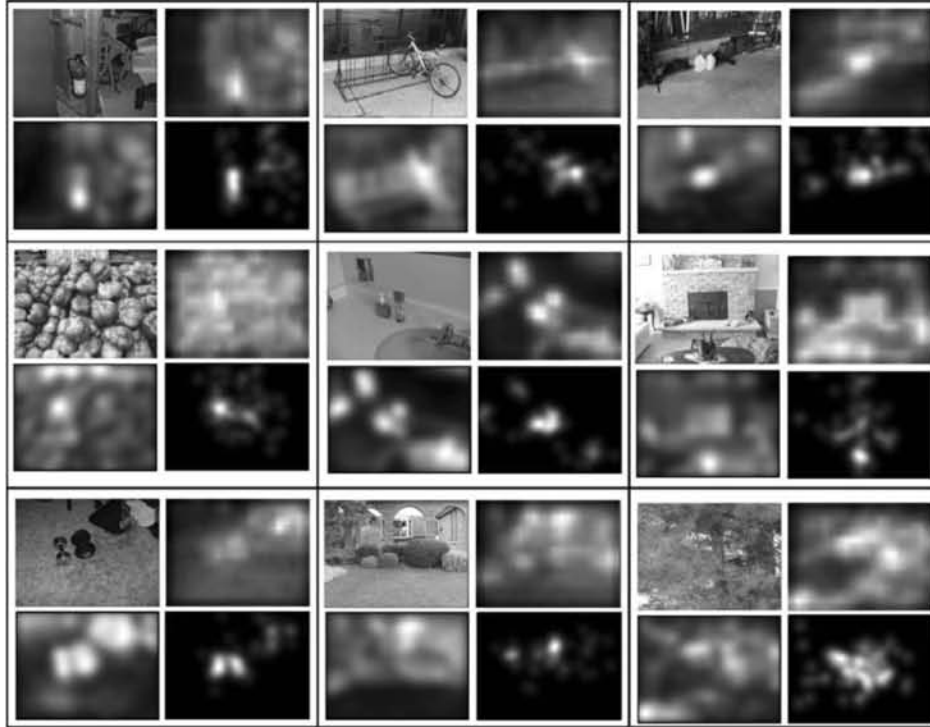

Figure 4: Results for qualitative comparison. Within each boxed region defined by solid lines: (Top Left) Original Image (Top Right) Saliency map produced by Itti + Koch algorithm. (Bottom Left) Saliency map based on information maximization. (Bottom Right) Fixation density map based on experimental human eye tracking data.

## 4   On Biological Plausibility

Although the proposed approach, along with the model of Itti and Koch describe saliency on the basis of a single topographical saliency map, there is mounting evidence that saliency in the primate brain is represented at several levels based on a hierarchical representation [18] of visual content. The proposed approach may accommodate such a configuration with the single necessary condition being a sparse representation at each layer.

As we have described in section 2, there is evidence that suggests the possibility that the primate visual system may consist of a multi-layer sparse coding architecture [10, 11]. The proposed algorithm quantifies information on the basis of a neural circuit, on units with response properties corresponding to neurons appearing in the primary visual cortex. However, given an analogous representation corresponding to higher visual areas that encode form, depth, convexity etc. the proposed method may be employed without any modification. Since the *popout* of features can occur on the basis of more complex properties such as a convex surface among concave surfaces [19], this is perhaps the next stage in a system that encodes saliency in the same manner as primates. Given a multi-layer architecture, the mechanism for selecting the locus of attention becomes less clear. In the model of Itti and Koch, a multi-layer winner-take-all network acts directly on the saliency map and there is no hierarchical representation of image content. There are however attention models that subscribe to a distributed representation of saliency (e.g. [20]), that may implement attentional selection with the proposed neural circuit encoding saliency at each layer.

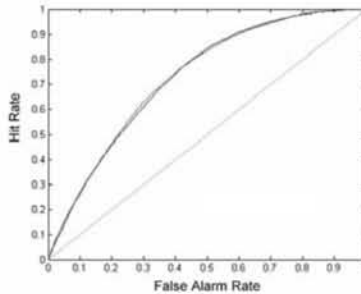

Figure 5: ROC curves for Self-information (blue) and Itti and Koch (red) saliency maps. Area under curves is 0.7288 and 0.7277 respectively.

## 5  Conclusion

We have described a strategy that predicts human attentional deployment on the principle of maximizing information sampled from a scene. Although no computational machinery is included strictly on the basis of biological plausibility, nevertheless the formulation results in an implementation based on a neurally plausible circuit acting on units that resemble those that facilitate early visual processing in primates. Comparison with an existing attention model reveals the efficacy of the proposed model in predicting salient image content. Finally, we demonstrate that the proposal might be generalized to facilitate selection based on high-level features provided an appropriate sparse representation is available.

## References

[1]  G.T. Buswell, How people look at pictures. Chicago: The University of Chicago Press.

[2]  A. Yarbus, Eye movements and vision. New York: Plenum Press.

[3]  L. Itti, C. Koch, E. Niebur, IEEE T PAMI, 11:1254-1259, 1998.

[4]  C. M. Privitera and L.W. Stark, IEEE T PAMI 22:970-981, 2000.

[5]  F. Fritz, C. Seifert, L. Paletta, H. Bischof, Proc. WAPCV, Graz, Austria, 2004.

[6]  L.W. Renninger, J. Coughlan, P. Verghese, J. Malik, Proceedings NIPS 17, Vancouver, 2004.

[7]  T. Kadir, M. Brady, IJCV 45(2):83-105, 2001.

[8]  T.S. Lee, S. Yu, Advances in NIPS 12:834-840 , Ed. S.A. Solla, T.K. Leen, K. Muller, MIT Press.

[9]  C. E. Shannon, The Bell Systems Technical Journal, 27:93-154, 1948.

[10]  D.J. Field, and B. A. Olshausen, Nature 381:607-609, 1996.

[11]  A.J. Bell, T.J. Sejnowski, Vision Research 37:3327-3338, 1997.

[12]  N. Bruce, Neurocomputing, 65-66:125-133, 2005.

[13]  P. Comon, Signal Processing 36(3):287-314, 1994.

[14]  M.W. Cannon and S.C. Fullenkamp, Vision Research 36(8):1115-1125, 1996.

[15]  B.W. Tatler, R.J. Baddeley, I.D. Gilchrist, Vision Research 45(5):643-659, 2005.

[16]  H. Koesling, E. Carbone, H. Ritter, University of Bielefeld, Technical Report, 2002.

[17]  T.W. Lee, M. Girolami, T.J. Sejnowski, Neural Computation 11:417-441, 1999.

[18]  J. Braun, C. Koch, D. K. Lee, L. Itti, In: Visual Attention and Cortical Circuits, (J. Braun, C. Koch, J. Davis Ed.), 215-242, Cambridge, MA:MIT Press, 2001.

[19]  J. Hullman, W. Te Winkel, F. Boselie, Perception and Psychophysics 62:162-174, 2000.

[20]  J.K. Tsotsos, S. Culhane, W. Wai, Y. Lai, N. Davis, F. Nuflo, Art. Intell. 78(1-2):507-547, 1995.
